# Plasticity Kernels and Temporal Statistics

Peter Dayan[1]    Michael Häusser[2]    Michael London[1,2]
[1]GCNU, [2]WIBR, Dept of Physiology
UCL, Gower Street, London
dayan@gatsby.ucl.ac.uk    {m.hausser,m.london}@ucl.ac.uk

## Abstract

Computational mysteries surround the kernels relating the magnitude and sign of changes in efficacy as a function of the time difference between pre- and post-synaptic activity at a synapse. One important idea[34] is that kernels result from *filtering, ie* an attempt by synapses to eliminate noise corrupting learning. This idea has hitherto been applied to trace learning rules; we apply it to experimentally-defined kernels, using it to reverse-engineer assumed signal statistics. We also extend it to consider the additional goal for filtering of weighting learning according to statistical surprise, as in the Z-score transform. This provides a fresh view of observed kernels and can lead to different, and more natural, signal statistics.

## 1 Introduction

Speculation and data that the rules governing synaptic plasticity should include a very special role for time[1,7,13,17,20,21,23,24,26,27,31,32,35] was spectacularly confirmed by a set of highly influential experiments[4,5,11,16,25] showing that the precise relative timing of pre-synaptic and post-synaptic action potentials governs the magnitude and sign of the resulting plasticity. These experimentally-determined rules (usually called spike-time dependent plasticity or STDP rules), which are constantly being refined,[18,30] have inspired substantial further theoretical work on their modeling and interpretation.[2,9,10,22,28,29,33] Figure 1(D1-G1)* depict some of the main STDP findings,[2] of which the best-investigated are shown in figure 1(D1;E1), and are variants of a 'standard' STDP rule. Earlier work considered *rate*-based rather than *spike*-based temporal rules, and so we adopt the broader term 'time dependent plasticity' or TDP. Note the strong temporal asymmetry in both the standard rules.

Although the theoretical studies have provided us with excellent tools for modeling the detailed consequences of different time-dependent rules, and understanding characteristics such as long-run stability and the relationship with non-temporal learning rules such as BCM,[6] specifically *computational* ideas about TDP are rather thinner on the ground. Two main qualitative notions explored in various of the works cited above are that the temporal asymmetries in TDP rules are associated with causality or prediction. However, looking specifically at the standard STDP rules, models interested in prediction

concentrate mostly on the LTP component and have difficulty explaining the precisely-timed nature of the LTD. Why should it be particularly detrimental to the weight of a synapse that the pre-synaptic action potential comes *just* after a post-synaptic action-potential, rather than 200ms later, for instance?. In the case of time-difference or temporal difference rules,[29,32] why might the LTD component be so different from the mirror reflection of the LTP component (figure 1(E1)), at least short of being tied to some particular biophysical characteristic of the post-synaptic cell. We seek alternative computationally-sound interpretations.

Wallis & Baddeley[34] formalized the intuition underlying one class of TDP rules (the so-called trace based rules, figure 1(A1)) in terms of temporal filtering. In their model, the actual output is a noisy version of a 'true' underlying signal. They suggested, and showed in an example, that learning proceeds more proficiently if the output is filtered by an optimal noise-removal filter (in their case, a Wiener filter) before entering into the learning rule. This is like using a prior over the signal, and performing learning based on the (mean) of the posterior over the signal given the observations (*ie* the output). If objects in the world normally persist for substantial periods, then, under some reasonable assumptions about noise, it turns out to be appropriate to apply a low-pass filter to the output. One version of this leads to a trace-like learning rule.

Of course, as seen in column 1 of figure 1, TDP rules are generally not trace-like. Here, we extend the Wallis-Baddeley (WB) treatment to rate-based versions of the actual rules shown in the figure. We consider two possibilities, which infer optimal signal models from the rules, based on two different assumptions about their computational role. One continues to regard them as Wiener filters. The other, which is closely related to recent work on adaptation and modulation,[3,8,15,36] has the kernel normalize frequency components according to their standard deviations, as well as removing noise. Under this interpretation, the learning signal is a Z-score-transformed version of the output.

In section 2, we describe the WB model. In section 3, we extend this model to the case of the observed rules for synaptic plasticity.

## 2 Filtering

Consider a set of pre-synaptic inputs $i \in \{1 \ldots n\}$ with firing rates $x_i(t)$ at time $t$ to a neuron with output rate $y(t)$. A general TDP plasticity rule suggests that synaptic weight $w_i$ should change according to the correlation between input $x_i(t)$ and output $y(t)$, through the medium of a temporal filter $\phi(s)$

$$\Delta w_i \propto \int dt x_i(t) \left\{ \int dt' y(t') \phi(t-t') \right\} = \int dt' y(t') \left\{ \int dt x_i(t) \phi(t-t') \right\} \quad (1)$$

Provided the temporal filters for each synapse on a single post-synaptic cell are the same, equation 1 indicates that pre-synaptic and post-synaptic filtering have essentially the same effect.

WB[34] consider the case that the output can be decomposed as $y(t) = s(t) + n(t)$, where $s(t)$ is a 'true' underlying *signal* and $n(t)$ is *noise* corrupting the signal. They suggest defining the filter so that $\hat{s}(t) = \int dt' y(t') \phi(t-t')$ is the optimal least-squares estimate of the signal. Thus, learning would be based on the best available information about the signal $s(t)$. If signal and noise are statistically stationary signals, with power spectra $|S(\omega)|^2$ and $|N(\omega)|^2$ respectively at (temporal) frequency $\omega$, then the magnitude of the Fourier transform

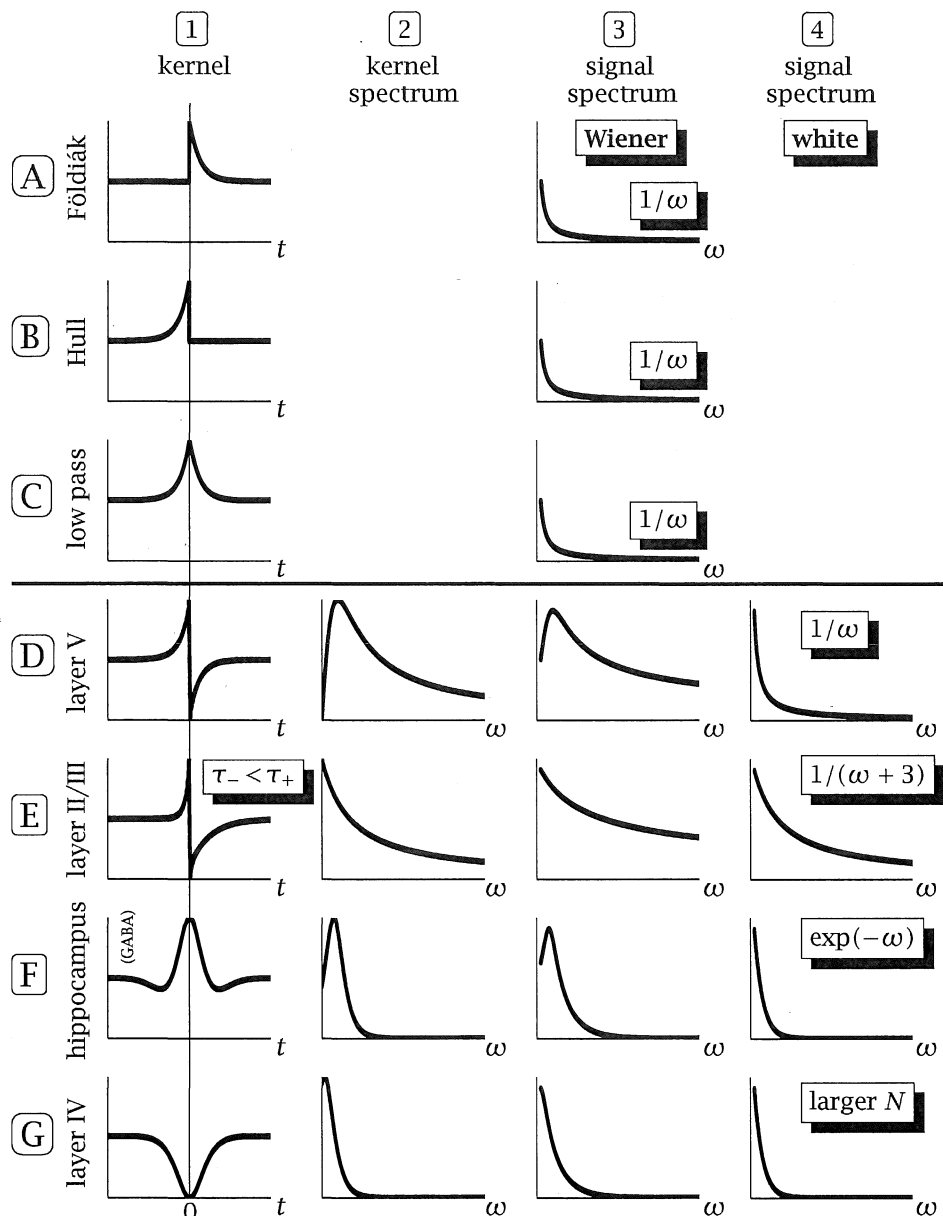

**Figure 1**: Time-dependent plasticity rules. The *rows* are for various suggested rules (A;[17] B;[23] D;[25] E;[16] F;[2] G,[14] from Abbott & Nelson[2]); the *columns* show: (1) the kernels in time $t$; (2) their temporal power spectra as a function of frequency $\omega$; (3) signal power $S(\omega)$ as function of $\omega$ assuming the kernels are derived from the underlying Wiener filter; (4) signal power $S(\omega)$ assuming the kernels are derived from the noise-removal and whitening filter. Different kernels also have different phase spectra. See text for more details. The ordinates of the plots have been individually normalized; but the abscissæ for all the temporal ($t$) plots and, separately, all the the spectral ($\omega$) plots, are the same, for the purposes of comparison. Numerical scales are omitted to focus on structural characteristics. In the text, we refer to individual graphs in this figure by their row letter (A-G) and column number (1-4).

of the (Wiener) filter is

$$|\Phi(\omega)| = \frac{|S(\omega)|^2}{|S(\omega)|^2 + |N(\omega)|^2} \tag{2}$$

Any filter with this spectral tuning will eliminate noise as best as possible; the remaining freedom lies in choosing the phases of the various frequency components. Following Földiák,[17] WB suggest using a causal filter for $y(t)$, with $\phi(t - t') = 0$ for $t < t'$. This means that the input $x_i(t)$ at time $t$ is correlated with weighted values of $y(t')$ for times $t' \leq t$ only. In fact, WB derive the optimal *acausal* filter and take its casual half, which is not necessarily the same thing. Interestingly, the forms of TDP that have commonly been used in reinforcement learning[23,31,32] consider purely acausal filters for $y(t)$ (such that $x_i(t)$ is correlated with *future* values of the output), and therefore use exactly the opposite condition on the filter, namely that $\phi(t - t') = 0$ for $t > t'$.

In the context of input coming from visually presented objects, WB suggest using white noise $N(\omega) = N, \forall \omega$, and consider two possibilities for $S(\omega)$, based on the assumptions that objects persist for either fixed, or randomly variable, lengths of time. We summarize their main result in the first three rows of figure 1. Figure 1(A3) shows the assumed, scale-free, magnitude spectrum $|S(\omega)| = 1/\omega$ for the signal. Figure 1(A1) shows the (truly optimal) purely causal version of the filter that results – it can be shown to involve exactly an exponential decay, with a rate constant which depends on the level of the noise $N$. In WB's self-supervised setting, it is rather unclear *a priori* whether the assumption of white noise is valid; WB's experiments bore it out to a rough approximation, and showed that the filter of figure 1(A1) worked well on a task involving digit representation and recognition.

Figure 1(B1;B3) repeat the analysis, with the same signal spectrum, but for the optimal purely acausal filter as used in reinforcement learning's synaptic eligibility traces. Of course, the true TDP kernels (shown in figure 1(D1-G1)) are neither purely casual nor acausal; figure 1(C1) shows the normal low pass filter that results from assuming phase 0 for all frequency components.

Although the WB filter of figure 1(C1) somewhat resembles a Hebbian version of the anti-Hebbian rule for layer IV spiny stellate cells shown in figure 1(G1), it is clearly not a good match for the standard forms of TDP. One might also question the relationship between the time constants of the kernels and the signal spectrum that comes from object persistence. The next section considers two alternative possibilities for interpreting TDP kernels.

# 3   Signalling and Whitening

The main intent of this paper is to combine WB's idea about the role of filtering in synaptic plasticity with the actual forms of the kernels that have been revealed in the experiments. Under two different models for the computational goal of filtering, we work back from the experimental kernels to the implied forms of the statistics of the signals. The first method employs WB's Wiener filtering idea. The second method can be seen as using a more stringent defintion of statistical significance.

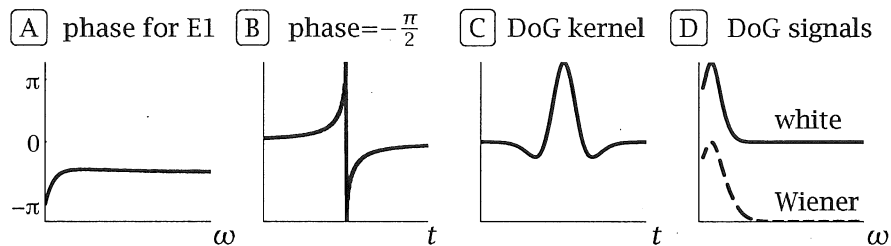

| A | phase for E1 | B | phase=$-\frac{\pi}{2}$ | C | DoG kernel | D | DoG signals |

**Figure 2:** Kernel manipulation. A) The phase spectrum (*ie* kernel phase as a function of frequency) for the kernel (shown in figure 1(E1)) with asymmetric LTP and LTD.[16] B) The kernel that results from the power spectrum of figure 1(E2) but constant phase $-\pi/2$. This kernel has symmetric LTP and LTD, with an intermediate time constant. C) Plasticity kernel that is exactly a difference of two Gaussians (DoG; compare figure 1(F1)). White (solid; from equation 4) and Wiener (dashed; from equation 3) signal spectra derived from the DoG kernel in (C). Here, the signal spectrum in the case of whitening has been vertically displaced so it is clearer. Both signal spectra show clear periodicities.

## 3.1 Reverse engineering signals from Wiener filtering

Accepting equation 2 as the form of the filter (note that this implies that $|\Phi(\omega)| \leq 1$), and, with WB, making the assumption that the noise is white, so $|N(\omega)| = N, \forall \omega$, the assumed amplitude spectrum of the signal process $s(t)$ is

$$|S(\omega)| = N\sqrt{|\Phi(\omega)|/(1 - |\Phi(\omega)|)} \ . \tag{3}$$

Importantly, the assumed power of the noise does *not* affect the form of the signal power, it only scales it.

Figure 1(D2-G2) shows the magnitude of the Fourier transform of the experimental kernels (which are shown in figure 1(D1-G1)), and figure 1(D3-G3) show the implied signal spectra. Since there is no natural data that specify the absolute scale of the kernels (*ie* the maximum value of $|\Phi(\omega)|$), we set it arbitrarily to 0.5. Any value less than $\sim 0.9$ leads to similar predictions for the signal spectra. We can relate figure 1(D3-G3) to to the heuristic criteria mentioned above for the signal power spectrum. In two cases (D3;F3), the clear peaks in the signal power spectra imply strong periodicities. For layer V pyramids (D3), the time constant for the kernel is $\sim 20$ms, implying a peak frequency of $\omega = 50$Hz in the $\gamma$ band. In the hippocampal case, the frequency may be a little lower. Certainly, the signal power spectra underlying the different kernels have quite different forms.

## 3.2 Reverse engineering signals from whitening

WB's suggestion that the underlying signal $s(t)$ should be extracted from the output $y(t)$ far from exhausts the possibilities for filtering. In particular, there have been various suggestions[36] that learning should be licensed by statistical *surprise, ie* according to how components of the output differ from expectations. A simple form of this that has gained recent currency is the Z-score transformation,[8,15,36] which implies considering components of the signal in units of (*ie* normalized by) their standard deviations. Mechanistically, this is closely related to whitening in the face of input noise, but with a rather different computational rationale.

A simple formulation of a noise-sensitive Z-score is Dong & Atick's[12] whitening filter. Under the same formulation as WB (equation 2), this suggests multiplying the Wiener filter by $1/|S(\omega)|$, giving

$$|\Phi(\omega)| = |S(\omega)|/(|S(\omega)|^2 + N(\omega)^2) . \qquad (4)$$

As in equation 3, it is possible to solve for the signal power spectra implied by the various kernels. The 4th column of figure 1 shows the result of doing this for the experimental kernels. In particular, it shows that the clear spectral peaks suggested by the Wiener filter (in the 3rd column) may be artefactual – they can arise from a form of whitening. Unlike the case of Wiener filtering, the signal statistics derived from the assumption of whitening have the common characteristic of monotonically decreasing signal powers as a function of frequency $\omega$, which is a common finding for natural scene statistics, for instance.

The case of the layer V pyramids[25] (row D in figure 1) is particularly clear. If the time constants of potentiation (LTP) and depression (LTD) are $\tau$, and LTP and LTD are matched, then the Fourier transform of the plasticity kernel is

$$\Phi(\omega) = \frac{1}{\sqrt{2\pi}} \left( \frac{1}{i\omega + \frac{1}{\tau}} + \frac{1}{i\omega - \frac{1}{\tau}} \right) = -i\sqrt{\frac{2}{\pi}} \frac{\omega}{\omega^2 + \frac{1}{\tau^2}} = -i\tau^2 \sqrt{\frac{2}{\pi}} \frac{\frac{1}{\omega}}{\frac{1}{\omega^2} + \tau^2} \qquad (5)$$

which is exactly the form of equation 4 for $S(\omega) = 1/\omega$ (which is duly shown in figure 1(D4)). Note the factor of $-i$ in $\Phi(\omega)$. This is determined by the phases of the frequency components, and comes from the anti-symmetry of the kernel. The phase of the components ($\angle\Phi(\omega) = -\pi/2$, by one convention) implies the predictive nature of the kernel: $x_i(t)$ is being correlated with led (*ie* future) values of noise-filtered, significance-normalized, outputs.

The other cases in figure 1 follow in a similar vein. Row E, from cortical layer II/II, with its asymmetry between LTP and LTD, has similar signal statistics, but with an extra falloff constant $\omega_0$, making $S(\omega) = 1/(\omega + \omega_0)$. Also, it has a phase spectrum $\angle\Phi(\omega)$ which is not constant with $\omega$ (see figure 2A). Row F, from hippocampal GABAergic cells in culture, has a form that can arise from an exponentially decreasing signal power and little assumed noise (small $N(\omega)$). Conversely, row G, in cortical layer IV spiny-stellate cells, arises from the same signal statistics, but with a large noise term $N(\omega)$. Unlike the case of the Wiener filter (equation 3), the form of the signal statistics, and not just their magnitude, depends on the amount of assumed noise.

Figure 2B-C show various aspects of how these results change with the parameters or forms of the kernels. Figure 2B shows that coupling the power spectrum (of figure 1E2) for the rule with asymmetric LTP and LTD with a constant phase spectrum ($-\pi/2$) leads to a rule with the same filtering characteristic, but with symmetric LTP and LTD. The phase spectrum concerns the predictive relationship between pre- and post-synaptic frequency components; it will be interesting to consider the kernels that result from other temporal relationships between pre- and post-synaptic activities. Figure 2C shows the kernel generated as a difference of two Gaussians (DoG). Although this kernel resembles that of figure 1F1, the signal spectra (figure 2D) calculated on the basis of whitening (solid; vertically displaced) or Wiener filtering (dashed) are similar to each other, and both involve strong periodicity near the spectral peak of the kernel.

# 4 Discussion

Temporal asymmetries in synaptic plasticity have been irresistibly alluring to theoretical treatments. We followed the suggestion that the kernels indicate that learning is *not* based on simple correlation between pre- and post-synaptic activity, but rather involves filtering in the light of prior information, either to remove noise from the signals (Wiener filtering), or to remove noise *and* boost components of the signals according to their statistical significance.

Adopting this view leads to new conclusions about the kernels, for instance revealing how the phase spectrum differentiates rules with symmetric and asymmetric potentiation and depression components (compare figures 1(E1); 2B). Making some further assumptions about the characteristics of the assumed noise, it permits us to reverse engineer the assumed statistics of the signals, *ie* to give a window onto the priors at synapses or cells (columns 3;4 of figure 1). Structural features in these signal statistics, such as strong periodicities, may be related to experimentally observable characteristics such as oscillatory activity in relevant brain regions. Most importantly, on this view, the detailed characteristics of the filtering might be expected to *adapt* in the light of patterns of activity. This suggests the straightforward experimental test of manipulating the input and/or output statistics and recording the consequences.

Various characteristics of the rules bear comment. Since we wanted to focus on structural features of the rules, the graphs in the figures all lack precise time or frequency scales. In some cases we know the time constants of the kernels, and they are usually quite fast (on the order of tens of milliseconds). This can suggest high frequency spectral peaks in assumed signal statistics. However, it also hints at the potential inadequacy of our rate-based treatment that we have given, and suggests the importance of a spike-based treatment.[22,30] Recent evidence that successive pairs of pre- and post-synaptic spikes do not interact additively in determining the magnitude and direction of plasticity[18] make the averaging inherent in the rate-based approximation less appealing. Further, we commented at the outset that pre- and post-synaptic filtering have similar effects, provided that all the filters on one post-synaptic cell are the same. If they are different, then synapses might well be treated as individual filters, ascertaining important signals for learning. In our framework, it is interesting to speculate about the role of (pre-)synaptic depression itself as a form of noise filter (since noise should be filtered before it can affect the *activity* of the post-synaptic cell, rather than just its plasticity); leaving the kernel as a significance filter, as in the whitening treatment. Finally, largely because of the separate roles of signal and noise, we have been unable to think of a simple experiment that would test between Wiener and whitening filtering. However, it is a quite critical issue in further exploring computational accounts of plasticity.

## Acknowledgements

We are very grateful to Odelia Schwartz for helpful discussions. Funding was from the Gatsby Charitable Foundation, the Wellcome Trust (MH) and an HFSP Long Term Fellowship (ML).

## Footnotes

*We refer to graphs in this figure by row and column.

# References

[1] Abbott, LF, & Blum, KI (1996) Functional significance of long-term potentiation for sequence learning and prediction. *Cerebral Cortex* 6:406–416.

[2] Abbott, LF & Nelson, SB (2000) Synaptic plasticity: taming the beast. *Nature Neuroscience* 3:1178-1183.

[3] Atick, JJ, Li, Z, & Redlich, AN (1992) Understanding retinal color coding from first principles. *Neural Computation* 4:559–572.

[4] Bell, CC, Han, VZ, Sugawara, Y & Grant K (1997) Synaptic plasticity in a cerebellum-like structure depends on temporal order. *Nature* 387:278-81.

[5] Bi, GQ & Poo, MM (1998) Synaptic modifications in cultured hippocampal neurons: dependence on spike timing, synaptic strength, and postsynaptic cell type. *Journal of Neuroscience* 18:10464-10472.

[6] Bienenstock, EL, Cooper, LN, & Munro, PW (1982) Theory for the development of neuron selectivity: Orientation specificity and binocular interaction in visual cortex. *Journal of Neuroscience* 2:32-48.

[7] Blum, KI, & Abbott, LF (1996) A model of spatial map formation in the hippocampus of the rat. *Neural Computation* 8:85-93.

[8] Buiatti, M & Van Vreeswijk, C (2003) Variance normalisation: a key mechanism for temporal adaptation in natural vision? *Vision Research*, in press.

[9] Cateau, H & Fukai, T (2003) A stochastic method to predict the consequence of arbitrary forms of spike-timing-dependent plasticity. *Neural Computation* 15:597-620.

[10] Chechik, G (2003). Spike time dependent plasticity and information maximization. *Neural Computation* in press.

[11] Debanne, D, Gahwiler, BH & Thompson, SM (1998) Long-term synaptic plasticity between pairs of individual CA3 pyramidal cells in rat hippocampal slice cultures. *Journal of Physiology* 507:237-247.

[12] Dong, DW, & Atick, JJ (1995) Temporal decorrelation: A theory of lagged and nonlagged responses in the lateral geniculate nucleus. *Network: Computation in Neural Systems* 6:159–178.

[13] Edelman, S & Weinshall, D (1991) A self-organizing multiple-view representation of 3D objects. *Biological Cybernetics* 64:209-219.

[14] Egger, V, Feldmeyer, D & Sakmann, B (1999) Coincidence detection and changes of synaptic efficacy in spiny stellate neurons in rat barrel cortex. *Nature Neuroscience* 2:1098-1105.

[15] Fairhall, AL, Lewen, GD, Bialek, W & de Ruyter Van Steveninck, RR (2001) Efficiency and ambiguity in an adaptive neural code. *Nature* 412:787-792.

[16] Feldman, DE (2000) Timing-based LTP and LTD at vertical inputs to layer II/III pyramidal cells in rat barrel cortex. *Neuron* 27:45-56.

[17] Földiák, P (1991) Learning invariance from transformed sequences. *Neural Computation* 3:194–200.

[18] Froemke, RC & Dan, Y (2002) Spike-timing-dependent synaptic modification induced by natural spike trains. *Nature* 416:433-438.

[19] Ganguly K, Kiss, L & Poo, M (2000) Enhancement of presynaptic neuronal excitability by correlated presynaptic and postsynaptic spiking. *Nature Neuroscience* 3:1018-1026.

[20] Gerstner, W & Abbott, LF (1997) Learning navigational maps through potentiation and modulation of hippocampal place cells. *Journal of Computational Neuroscience* 4:79-94.

[21] Gerstner, W, Kempter, R, van Hemmen, JL & Wagner, H (1996) A neuronal learning rule for sub-millisecond temporal coding. *Nature* 383:76-81.

[22] Gerstner, W & Kistler, WM (2002) Mathematical formulations of Hebbian learning. *Biological Cybernetics* 87:404-15.

[23] Hull, CL (1943) *Principles of Behavior* New York, NY: Appleton-Century.

[24] Levy, WB & Steward, D (1983) Temporal contiguity requirements for long-term associative potentiation/depression in the hippocampus. *Neuroscience* 8:791-797

[25] Markram, H, Lubke, J, Frotscher, M, & Sakmann, B (1997) Regulation of synaptic efficacy by coincidence of postsynaptic APs and EPSPs. *Science* 275:213-215.

[26] Minai, AA, & Levy, WB (1993) Sequence learning in a single trial. *International Neural Network Society World Congress of Neural Networks II*. Portland, OR: International Neural Network Society, 505–508.

[27] Pavlov, PI (1927) *Conditioned Reflexes* Oxford, England, OUP.

[28] Porr, B & Wörgötter, F (2003) Isotropic sequence order learning. *Neural Computation* 15:831-864.

[29] Rao, RP & Sejnowski, TJ (2001) Spike-timing-dependent Hebbian plasticity as temporal difference learning. *Neural Computation* 13:2221-2237.

[30] Sjöström, PJ, Turrigiano, GG & Nelson, SB (2001) Rate, timing, and cooperativity jointly determine cortical synaptic plasticity. *Neuron* 32:1149-1164.

[31] Sutton, RS (1988) Learning to predict by the methods of temporal difference. *Machine Learning* 3:9-44.

[32] Sutton, RS & Barto, AG (1981) Toward a modern theory of adaptive networks: Expectation and prediction. *Psychological Review* 88:135-170.

[33] van Rossum, MC, Bi, GQ & Turrigiano, GG (2000) Stable Hebbian learning from spike timing-dependent plasticity. *Journal of Neuroscience* 20:8812-21.

[34] Wallis, G & Baddeley, R (1997) Optimal, unsupervised learning in invariant object recognition. *Neural Computation* 9:883-894.

[35] Wallis, G & Rolls, ET (1997). Invariant face and object recognition in the visual system. it Progress in Neurobiology 51:167-194.

[36] Yu, AJ & Dayan, P (2003) Expected and unexpected uncertainty: ACh & NE in the neocortex. In *NIPS 2002* Cambridge, MA: MIT Press.
